# Collaborative Gaussian Processes for Preference Learning

**Neil Houlsby** *
Department of Engineering
University of Cambridge

**Jose Miguel Hernández-Lobato** *
Department of Engineering
University of Cambridge

**Ferenc Huszár**
Department of Engineering
University of Cambridge

**Zoubin Ghahramani**
Department of Engineering
University of Cambridge

## Abstract

We present a new model based on Gaussian processes (GPs) for learning pairwise preferences expressed by multiple users. Inference is simplified by using a *preference kernel* for GPs which allows us to combine supervised GP learning of user preferences with unsupervised dimensionality reduction for multi-user systems. The model not only exploits collaborative information from the shared structure in user behavior, but may also incorporate user features if they are available. Approximate inference is implemented using a combination of expectation propagation and variational Bayes. Finally, we present an efficient active learning strategy for querying preferences. The proposed technique performs favorably on real-world data against state-of-the-art multi-user preference learning algorithms.

## 1   Introduction

Preference learning is concerned with making inference from data consisting of pairs of items and corresponding binary labels indicating user preferences. This data arises in many contexts, including medical assistive technologies [1], graphical design [3] and recommendation systems [5]. A popular modeling approach assumes the existence of a utility function $f$ such that $f(\mathbf{x})$ gives the utility of an item with feature vector $\mathbf{x}$; $f(\mathbf{x}_i) > f(\mathbf{x}_j)$ indicates that item $i$ is preferred to item $j$. Bayesian methods can be used to learn $f$, for example, by modeling $f$ independently for each user as a draw from a Gaussian process (GP) prior [4]. However, when data from many users is available, such methods do not leverage similarities in preferences across users. Current multi-user approaches require that features are available for each user and assume that users with similar features have similar preferences [2], or perform single-user learning, ignoring user features, but tie information across users with a hierachical prior [1]. These methods are not flexible and can only address one of two possible scenarios: a) user features are available and they are useful for prediction and b) when this is not the case. Additionally, they involve at least solving $U$ GP problems, where $U$ is the total number of users. This cost is prohibitive even for modest $U$. Our approach, by contrast, can address both a) and b) by combining informative user features with collaborative information. Furthermore, we perform scalable inference which can handle problems with large $U$.

Our new multi-user model is based on dimensionality reduction ideas from the field of collaborative filtering [19, 16]. Unsupervised learning of similarities in users' behavior is exploited without requiring access to user-specific feature vectors. However, if these are available it may be desirable

---

to incorporate them for predictions; our model can use these user-specific features as well. The proposed method is based on a connection between preference learning and GP binary classification. We show that both problems are equivalent when a covariance function called the *preference kernel* is used. This specific kernel simplifies the inference process, allowing us to implement more complex models such as the proposed multi-user approach. Finally, in real scenarios, querying users for preference may be costly and intrusive, so it is desirable to learn preferences using the least data possible. With this objective, we present BALD (Bayesian active learning by disagreement), an efficient active learning strategy for binary classification problems with GP priors.

## 2 Pairwise preference learning as special case of binary classification

The problem of pairwise preference learning can be recast as a special case of binary classification. Let us consider two items $i$ and $j$ with corresponding feature vectors $\mathbf{x}_i, \mathbf{x}_j \in \mathcal{X}$. In the pairwise preference learning problem, we are given pairs of feature vectors $\mathbf{x}_i$ and $\mathbf{x}_j$ and corresponding class labels $y \in \{-1, 1\}$ such that $y = 1$ if the user prefers item $i$ to item $j$ and $y = -1$ otherwise. The task of interest is then to predict the class label for a new pair of feature vectors not seen before. This problem can be addressed by introducing a latent preference function $f : \mathcal{X} \mapsto \mathbb{R}$ such that $f(\mathbf{x}_i) > f(\mathbf{x}_j)$ whenever the user prefers item $i$ to item $j$ and $f(\mathbf{x}_i) < f(\mathbf{x}_j)$ otherwise [4]. When the evaluations of $f$ are contaminated with Gaussian noise with zero mean and (without loss of generality) variance $1/2$, we obtain the following likelihood for $f$ given $\mathbf{x}_i, \mathbf{x}_j$ and $y$

$$\mathcal{P}(y|\mathbf{x}_i, \mathbf{x}_j, f) = \Phi[(f[\mathbf{x}_i] - f[\mathbf{x}_j])y]\,, \tag{1}$$

where $\Phi$ is the standard Gaussian cumulative distribution function. The preference learning problem can be solved by combining a GP prior on $f$ with the likelihood function in (1) [4]. The posterior for $f$ can then be used to make predictions on the user preferences for new pairs of items.

Note that the likelihood (1) depends only on the difference between $f(\mathbf{x}_i)$ and $f(\mathbf{x}_j)$. Let $g : \mathcal{X}^2 \mapsto \mathbb{R}$ be the latent function $g(\mathbf{x}_i, \mathbf{x}_j) = f(\mathbf{x}_i) - f(\mathbf{x}_j)$. We can recast the inference problem in terms of $g$ and ignore $f$. When the evaluation of $g$ is contaminated with standard Gaussian noise, the likelihood for $g$ given $\mathbf{x}_i, \mathbf{x}_j$ and $y$ is

$$\mathcal{P}(y|\mathbf{x}_i, \mathbf{x}_j, g) = \Phi[g(\mathbf{x}_i, \mathbf{x}_j)y]\,. \tag{2}$$

Since $g$ is obtained from $f$ through a linear operation, the GP prior on $f$ induces a GP prior on $g$. The covariance function $k_{\text{pref}}$ of the GP prior on $g$ can be computed from the covariance function $k$ of the GP on $f$ as $k_{\text{pref}}((\mathbf{x}_i, \mathbf{x}_j), (\mathbf{x}_k, \mathbf{x}_l)) = k(\mathbf{x}_i, \mathbf{x}_k) + k(\mathbf{x}_j, \mathbf{x}_l) - k(\mathbf{x}_i, \mathbf{x}_l) - k(\mathbf{x}_j, \mathbf{x}_k)$. The derivations can be found in Section 1 of the supplementary material. We call $k_{\text{pref}}$ the *preference kernel*. The same kernel function can be derived from a large margin classification viewpoint [6]. However, to our knowledge, the preference kernel has not been used previously for GP-based models.

The combination of (2) with a GP prior based on the preference kernel allows us to transform the pairwise preference learning problem into binary classification with GPs. This means that state-of-the-art methods for GP binary classification, such as expectation propagation [14], can be applied directly to preference learning. Furthermore, the simplified likelihood (2) allows us to implement complex methods such as the multi-user approach which is described in the following section.

## 3 Multi-user preference learning

Consider $I$ items with feature vectors $\mathbf{x}_i \in \mathcal{X}$ for $i = 1, \ldots, I$. The single-user learning approach assumes an independent latent function for the $u$-th user, $g_u : \mathcal{X}^2 \mapsto \mathbb{R}$. Our approach to the multi-user problem is to assume common structure in the user latent functions. In particular, we assume a set of $D$ shared latent functions, $h_d : \mathcal{X}^2 \mapsto \mathbb{R}$ for $d = 1, \ldots, D$, such that the user latent functions are generated by a linear combination of these functions, namely

$$g_u(\mathbf{x}_j, \mathbf{x}_k) = \sum_{d=1}^{D} w_{u,d} h_d(\mathbf{x}_j, \mathbf{x}_k)\,, \tag{3}$$

here $w_{u,d} \in \mathbb{R}$ is the weight given to function $h_d$ for user $u$. We place a GP prior over the shared latent functions $h_1, \ldots, h_D$ using the preference kernel described in the previous section. This model allows the preferences of the different users to share some common structure represented by the latent functions $h_1, \ldots, h_D$. This approach is similar to dimensionality reduction methods that are commonly used for addressing collaborative filtering problems [19, 16].

We may extend this model further to the case in which, for each user $u$, there is a feature vector $\mathbf{u}_u \in \mathcal{U}$ containing information that might be useful for prediction. We denote by $\mathbf{U}$ the set of all the users' feature vectors, that is, $\mathbf{U} = \{\mathbf{u}_1, \ldots, \mathbf{u}_U\}$. The user features are incorporated now by placing a separate GP prior over the users weights. In particular, we replace the scalars $w_{u,d}$ in (3) with functions $w'_d(\mathbf{u}_u) : \mathcal{U} \to \mathbb{R}$. These weight functions describe the contribution of shared latent function $h_d$ to the user latent function $g_u$ as a function of the user feature vector $\mathbf{u}_u$.

In the multi-user setting we are given a list $\mathcal{L} = \{p_1, \ldots, p_P\}$ with all the *pairs* of items evaluated by the users, where $P \leq I(I-1)/2$ (the maximum number of pairs). The data consists of $\mathcal{L}$, the sets of feature vectors for the users $\mathbf{U}$ (if available), the item features $\mathbf{X} = \{\mathbf{x}_1, \ldots, \mathbf{x}_I\}$, and $U$ sets of preference judgements, one for each user, $\mathcal{D} = \{\{z_{u,i}, y_{u,i}\}_{i=1}^{M_u}\}_{u=1}^{U}$, where $z_{u,i}$ indexes the $i$-th pair evaluated by user $u$, $y_{i,u} = 1$ if this user prefers the first item in the pair to the second and $y_{i,u} = -1$ otherwise. $M_u$ is the number of preference judgements made by the $u$-th user.

## 3.1 Probabilistic description

To address the task of predicting preference on unseen item pairs we cast the model into a probabilistic framework. Let $\mathbf{G}$ be an $U \times P$ 'user-function' matrix, where each row corresponds to a particular user's latent function, that is, the entry in the $u$-th column and $i$-th row is $g_{u,i} = g_u(\mathbf{x}_{\alpha(i)}, \mathbf{x}_{\beta(i)})$ and $\alpha(i)$ and $\beta(i)$ denote respectively the first and second item in the $i$-th pair from $\mathcal{L}$. Let $\mathbf{H}$ be a $D \times P$ 'shared-function' matrix, where each row represents the shared latent functions, that is, the entry in the $d$-th row and $i$-th column is $h_{d,i} = h_d(\mathbf{x}_{\alpha(i)}, \mathbf{x}_{\beta(i)})$. Finally, we introduce the $U \times D$ weight matrix $\mathbf{W}$ such that each row contains a user's weights, that is, the entry in the $u$-th row and $d$-th column of this matrix is $w'_d(\mathbf{u}_u)$. Note that $\mathbf{G} = \mathbf{W}\mathbf{H}$ represents equation (3) in matrix form. Let $\mathbf{T}$ be the $U \times P$ target matrix given by $\mathbf{T} = \text{sign}[\mathbf{G} + \mathbf{E}]$, where $\mathbf{E}$ is an $U \times P$ noise matrix whose entries are sampled i.i.d. from a standard Gaussian distribution and the function "sign" retains only the sign of the elements in a matrix. The observations $y_{u,i}$ in $\mathcal{D} = \{\{z_{u,i}, y_{u,i}\}_{i=1}^{M_u}\}_{u=1}^{U}$ are mapped to the corresponding entries of $\mathbf{T}$ using $t_{u,z_{u,i}} = y_{u,i}$. Let $\mathbf{T}^{(\mathcal{D})}$ and $\mathbf{G}^{(\mathcal{D})}$ represent the elements of $\mathbf{T}$ and $\mathbf{G}$ corresponding only to the available observations $y_{u,i}$ in $\mathcal{D}$. Then, the likelihood for $\mathbf{G}^{(\mathcal{D})}$ given $\mathbf{T}^{(\mathcal{D})}$ and conditional distribution for $\mathbf{G}^{(\mathcal{D})}$ given $\mathbf{H}$ and $\mathbf{W}$ are

$$\mathcal{P}(\mathbf{T}^{(\mathcal{D})}|\mathbf{G}^{(\mathcal{D})}) = \prod_{u=1}^{U}\prod_{i=1}^{M_u} \Phi[t_{u,z_{u,i}} g_{u,z_{u,i}}] \text{ and } \mathcal{P}(\mathbf{G}^{(\mathcal{D})}|\mathbf{W},\mathbf{H}) = \prod_{u=1}^{U}\prod_{i=1}^{M_u} \delta[g_{u,z_{u,i}} - \mathbf{w}_u \mathbf{h}_{\cdot,z_{u,i}}]$$

respectively, where $\mathbf{w}_u$ is the $u$-th row in $\mathbf{W}$, $\mathbf{h}_{\cdot,i}$ is the $i$-th column in $\mathbf{H}$ and $\delta$ represents a point probability mass at zero. We now select the priors for $\mathbf{W}$ and $\mathbf{H}$. We assume that each function $w'_1, \ldots, w'_D$ is sampled *a priori* from a GP with zero mean and specific covariance function. Let $\mathbf{K}_{\text{users}}$ be the $U \times U$ covariance matrix for entries in each column of matrix $\mathbf{W}$. Then

$$\mathcal{P}(\mathbf{W}|\mathbf{U}) = \prod_{d=1}^{D} \mathcal{N}(\mathbf{w}_{\cdot,d}|\mathbf{0}, \mathbf{K}_{\text{users}}), \tag{4}$$

where $\mathbf{w}_{\cdot,d}$ is the $d$-th column in $\mathbf{W}$. If user features are unavailable, $\mathbf{K}_{\text{users}}$ becomes the identity matrix. Finally, we assume that each shared latent function $h_1, \ldots, h_D$ is sampled *a priori* from a GP with zero mean and covariance function given by a preference kernel. Let $\mathbf{K}_{\text{items}}$ be the $P \times P$ preference covariance matrix for the item pairs in $\mathcal{L}$. The prior for $\mathbf{H}$ is then

$$\mathcal{P}(\mathbf{H}|\mathbf{X}, \mathcal{L}) = \prod_{j=1}^{D} \mathcal{N}(\mathbf{h}_j|\mathbf{0}, \mathbf{K}_{\text{items}}), \tag{5}$$

where $\mathbf{h}_j$ is the $j$-th row in $\mathbf{H}$. The resulting posterior for $\mathbf{W}$, $\mathbf{H}$ and $\mathbf{G}^{(\mathcal{D})}$ is

$$\mathcal{P}(\mathbf{W}, \mathbf{H}, \mathbf{G}^{(\mathcal{D})}|\mathbf{T}^{(\mathcal{D})}, \mathbf{X}, \mathcal{L}) = \frac{\mathcal{P}(\mathbf{T}^{(\mathcal{D})}|\mathbf{G}^{(\mathcal{D})})\mathcal{P}(\mathbf{G}^{(\mathcal{D})}|\mathbf{W},\mathbf{H})\mathcal{P}(\mathbf{W}|\mathbf{U})\mathcal{P}(\mathbf{H}|\mathbf{X},\mathcal{L})}{\mathcal{P}(\mathbf{T}^{(\mathcal{D})}|\mathbf{X},\mathcal{L})}. \tag{6}$$

Given a new item pair $p_{P+1}$, we can compute the predictive distribution for the preference of the $u$-th user ($1 \leq u \leq U$) on this pair by integrating out the parameters $\mathbf{H}$, $\mathbf{W}$ and $\mathbf{G}^{(\mathcal{D})}$ as follows:

$$\mathcal{P}(t_{u,P+1}|\mathbf{T}^{(\mathcal{D})}, \mathbf{X}, \mathcal{L}, p_{P+1}) = \int \mathcal{P}(t_{u,P+1}|g_{u,P+1})\mathcal{P}(g_{u,P+1}|\mathbf{w}_u, \mathbf{h}_{\cdot,P+1})$$

$$\mathcal{P}(\mathbf{h}_{\cdot,P+1}|\mathbf{H}, \mathbf{X}, \mathcal{L}, p_{P+1})\mathcal{P}(\mathbf{H}, \mathbf{W}, \mathbf{G}^{(\mathcal{D})}|\mathbf{T}^{(\mathcal{D})}, \mathbf{X}, \mathcal{L}) \, d\mathbf{H} \, d\mathbf{W} \, d\mathbf{G}^{(\mathcal{D})}, \tag{7}$$

where $\mathcal{P}(t_{u,P+1}|g_{u,P+1}) = \Phi[t_{u,P+1}g_{u,P+1}]$, $\mathcal{P}(g_{u,P+1}|\mathbf{w}_u, \mathbf{h}_{\cdot,P+1}) = \delta[g_{u,P+1} - \mathbf{w}_u \mathbf{h}_{\cdot,P+1}]$,

$$\mathcal{P}(\mathbf{h}_{\cdot,P+1}|\mathbf{H}, \mathbf{X}, \mathcal{L}, p_{P+1}) = \prod_{d=1}^{D} \mathcal{N}(h_{d,P+1}|\mathbf{k}_\star^\mathrm{T}\mathbf{K}_{\text{items}}^{-1}\mathbf{h}_d, k_\star - \mathbf{k}_\star^\mathrm{T}\mathbf{K}_{\text{items}}^{-1}\mathbf{k}_\star) \tag{8}$$

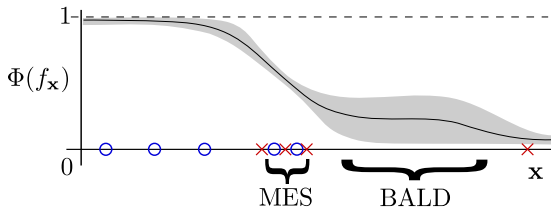

Figure 1: Toy example with 1D input. Circles and crosses denote labelled data. The plot shows the mean and variance of the GP predictive distribution. Maximum Entropy Sampling (MES) samples from the region of highest marginal uncertainty, ignoring the second term in (10). BALD samples from the region of greatest uncertainty in the latent function.

$k_\star$ is the prior variance of $h_d(\mathbf{x}_{\alpha(P+1)}, \mathbf{x}_{\beta(P+1)})$ and $\mathbf{k}_\star$ is a $P$-dimensional vector that contains the prior covariances between $h_d(\mathbf{x}_{\alpha(P+1)}, \mathbf{x}_{\beta(P+1)})$ and $h_d(\mathbf{x}_{\alpha(1)}, \mathbf{x}_{\beta(1)}), \ldots, h_d(\mathbf{x}_{\alpha(P)}, \mathbf{x}_{\beta(P)})$. Computing (6) or (8) is infeasible and approximations must be used. For this, we use a combination of expectation propagation (EP) [14] and variation Bayes (VB) [7]. Empirical studies show that EP obtains state-of-the-art performance in the related problem of GP binary classification [15].

We want to learn user preferences with the proposed model from the least amount of data possible. Therefore we desire to query users actively about their preferences on the most informative pairs of items [3]. Next, we describe a novel method to implement this strategy. This method exploits the preference kernel and so may be trivially generalized to GP binary classification problems also.

## 4 Bayesian active learning by disagreement

The goal of active learning is to choose item pairs such that we learn the preference functions for the users using minimal data. Information theoretic approaches to active learning are popular because they do not require prior knowledge of loss functions or test domains. The central goal is to identify the new data point that maximizes the expected reduction in posterior entropy. For preference learning (see Section 2), this implies choosing the new item features $\mathbf{x}_i$ and $\mathbf{x}_j$ that maximize

$$\mathrm{H}[\mathcal{P}(g|\mathcal{D})] - \mathbb{E}_{\mathcal{P}(y|\mathbf{x}_i,\mathbf{x}_j,\mathcal{D})}\left[\mathrm{H}[\mathcal{P}(g|y,\mathbf{x}_i,\mathbf{x}_j,\mathcal{D})]\right] , \qquad (9)$$

where $\mathcal{D}$ are the user preferences observed so far and $\mathrm{H}[p(x)] = -\int p(x)\log p(x)\,dx$ represents the Shannon entropy. This framework, originally proposed in [10], is difficult to apply directly to models based on GPs. In these models, entropies can be poorly defined or their computation can be intractable. In practice, current approaches make approximations for the computation of the posterior entropy [12, 9]. However, a second difficulty arises; if $n$ new data points are available for selection, with $|\{-1,1\}| = 2$ possible values for $y$. Then $\mathcal{O}(2n)$ potentially expensive posterior updates are required to find the maximizer of (9): one for every available feature vector and possible class value. This is often too expensive in practice.

A solution consists in noting that (9) is equivalent to the conditioned mutual information between $y$ and $g$. Using this we can rearrange this equation to compute entropies in $y$ space:

$$\mathrm{H}[\mathcal{P}(y|\mathbf{x}_i,\mathbf{x}_j,\mathcal{D})] - \mathbb{E}_{\mathcal{P}(g|\mathcal{D})}\left[\mathrm{H}\left[\mathcal{P}(y|\mathbf{x}_i,\mathbf{x}_j,g)\right]\right] . \qquad (10)$$

This overcomes the previous challenges. Entropies are now evaluated in output space, which has low dimension. Furthermore, $g$ is now conditioned only upon $\mathcal{D}$, so only $\mathcal{O}(1)$ updates of the posterior distribution are required. We only need to recompute the posterior once per data point selected, not for every possible data point under consideration. Expression (10) also provides us with an intuition about the objective; we seek the $\mathbf{x}_i$ and $\mathbf{x}_j$ for which a) the model is marginally uncertain about $y$ (high $\mathrm{H}[\mathcal{P}(y|\mathbf{x}_i,\mathbf{x}_j,\mathcal{D})]$) and b) conditioned on a particular value of $g$ the model is confident about $y$ (low $\mathbb{E}_{\mathcal{P}(g|\mathcal{D})}\left[\mathrm{H}[\mathcal{P}(y|\mathbf{x}_i,\mathbf{x}_j,g)]\right]$). This can be interpreted as seeking the pair $\mathbf{x}_i$ and $\mathbf{x}_j$ for which the latent functions $g$, under the posterior, 'disagree' with each other the most about the outcome, that is, the preference judgement. Therefore, we refer to this objective as Bayesian Active Learning by Disagreement (BALD). This method is independent of the approach used for inference, something which does not hold for the techniques described in [12, 8, 9]. In the following section we show how (10) can be applied to binary classification with GPs, and hence via the preference kernel also to any preference learning problem.

### 4.1 BALD in binary classification with GPs

Most approximate inference methods for the problem of binary classification with GPs produce a Gaussian approximation to the posterior distribution of $f$, the latent function of interest. In

the binary GP classifier, the entropy of $y$ given the corresponding value of $f$ can be expressed in terms of the binary entropy function, $\mathrm{h}[f] = -f \log f - (1 - f) \log(1 - f)$. In particular, $\mathrm{H}[p(y|\mathbf{x}, f)] = \mathrm{h}[\Phi(f(\mathbf{x}))]$. When a Gaussian is used to approximate the posterior of $f$, we have that for each $\mathbf{x}$, $f_{\mathbf{x}} = f(\mathbf{x})$ will follow a Gaussian distribution with mean $\mu_{\mathbf{x}}$ and variance $\sigma_{\mathbf{x}}^2$. The first term in (10), that is, $\mathrm{H}[p(y|\mathbf{x}, \mathcal{D})]$, can be handled analytically in this case: $\mathrm{H}[p(y|\mathbf{x}, \mathcal{D})] \approx \mathrm{h}\left[\int \Phi(f_{\mathbf{x}})\mathcal{N}(f_{\mathbf{x}}|\mu_{\mathbf{x}}, \sigma_{\mathbf{x}}^2)df_{\mathbf{x}}\right] = \mathrm{h}\left[\Phi\left(\mu_{\mathbf{x}}(\sigma_{\mathbf{x}}^2 + 1)^{-1/2}\right)\right]$, where $\approx$ represents here the Gaussian approximation to the posterior of $f_{\mathbf{x}}$. The second term in (10), that is, $\mathbb{E}_{p(f|\mathcal{D})}\left[\mathrm{H}[p(y|\mathbf{x}, f)]\right]$, can be approximated as $\mathbb{E}_{p(f|\mathcal{D})}\left[\mathrm{H}[p(y|\mathbf{x}, f)]\right] \approx C(\sigma_{\mathbf{x}}^2 + C^2)^{-1/2} \exp\left(-\mu_{\mathbf{x}}^2(2\left(\sigma_{\mathbf{x}}^2 + C^2\right))^{-1}\right)$, where $C = \sqrt{\pi \log 2/2}$. This result is obtained by using the Gaussian approximation to the posterior of $f_{\mathbf{x}}$ and then approximating $\mathrm{h}[\Phi(f_{\mathbf{x}})]$ by the squared exponential curve $\exp(-f_{\mathbf{x}}^2/\pi \log 2)$ (details can be found in Section 3 of the supplementary material).

To summarize, the BALD algorithm for active binary GP classification / preference learning first applies any approximate inference method to obtain the posterior mean $\mu_{\mathbf{x}}$ and variance $\sigma_{\mathbf{x}}^2$ of $f$ at each point of interest $\mathbf{x}$. Then, it selects the feature vector $\mathbf{x}$ that maximizes the objective

$$\mathrm{h}\left[\Phi\left(\mu_{\mathbf{x}}(\sigma_{\mathbf{x}}^2 + 1)^{-1/2}\right)\right] - C(\sigma_{\mathbf{x}}^2 + C^2)^{-1/2} \exp\left(-\mu_{\mathbf{x}}^2(2\left(\sigma_{\mathbf{x}}^2 + C^2\right))^{-1}\right). \tag{11}$$

BALD assigns a high value to the feature vector $\mathbf{x}$ when the model is both uncertain about the label ($\mu_{\mathbf{x}}$ close to 0) and there is high uncertainty about $f_{\mathbf{x}}$ ($\sigma_{\mathbf{x}}^2$ is large). The second term prevents BALD from sampling in regions where the model knows that the label is uncertain. Figure 1 illustrates the differences between BALD and Maximum Entropy Sampling [17] (details in the supplementary material, Section 5). MES considers only marginal uncertainty (the first term in (11)), and hence seeks data in an uninformative region of the plot. By contrast, BALD samples data from the region of greatest uncertainty in the latent function.

## 5 Expectation propagation and variational Bayes

Approximate inference in our model is implemented using a combination of expectation propagation (EP) [13] and variational Bayes (VB) [7]. Here, we briefly describe the method, but full details are in Section 4 of the supplementary. We approximate the posterior (6) by the parametric distribution

$$\mathcal{Q}(\mathbf{W}, \mathbf{H}, \mathbf{G}^{(\mathcal{D})}) = \left[\prod_{u=1}^{U}\prod_{d=1}^{D}\mathcal{N}(w_{ud}|m_{u,d}^w, v_{u,d}^w)\right]\left[\prod_{d=1}^{D}\prod_{i=1}^{P}\mathcal{N}(h_{d,i}|m_{d,i}^h, v_{d,i}^h)\right]$$
$$\left[\prod_{u=1}^{N}\prod_{j=1}^{M_u}\mathcal{N}(g_{u,z_{u,j}}|m_{u,j}^g, v_{u,j}^g)\right], \tag{12}$$

where $m_{u,d}^w$, $v_{u,d}^w$, $m_{d,i}^h$, $v_{d,i}^h$, $m_{u,j}^g$, and $v_{u,j}^g$ are free parameters to be determined by EP and the superscripts $w$, $h$ and $g$ indicate the random variables described by these parameters. The joint distribution $\mathcal{P}(\mathbf{G}^{(\mathcal{D})}, \mathbf{W}, \mathbf{H}, \mathbf{T}^{(\mathcal{D})}, \mathbf{X}, \ell)$ can be factorized into four factors $f_1, \ldots, f_4$, namely, $\mathcal{P}(\mathbf{G}^{(\mathcal{D})}, \mathbf{W}, \mathbf{H}, \mathbf{T}^{(\mathcal{D})}, \mathbf{X}, \ell) = \prod_{a=1}^{4} f_a(\mathbf{G}^{(\mathcal{D})}, \mathbf{W}, \mathbf{H})$, where $f_1(\mathbf{G}^{(\mathcal{D})}, \mathbf{W}, \mathbf{H}) = \mathcal{P}(\mathbf{T}^{(\mathcal{D})}|\mathbf{G}^{(\mathcal{D})})$, $f_2(\mathbf{G}^{(\mathcal{D})}, \mathbf{W}, \mathbf{H}) = \mathcal{P}(\mathbf{G}^{(\mathcal{D})}|\mathbf{W}, \mathbf{H})$, $f_3(\mathbf{G}^{(\mathcal{D})}, \mathbf{W}, \mathbf{H}) = \mathcal{P}(\mathbf{W}|\mathbf{U})$ and $f_4(\mathbf{G}^{(\mathcal{D})}, \mathbf{W}, \mathbf{H}) = \mathcal{P}(\mathbf{H}|\mathbf{X}, \ell)$. EP approximates these exact factors by approximate factors $\hat{f}_1(\mathbf{W}, \mathbf{H}, \mathbf{G}^{(\mathcal{D})}), \ldots, \hat{f}_4(\mathbf{W}, \mathbf{H}, \mathbf{G}^{(\mathcal{D})})$ that have the same functional form as $\mathcal{Q}$

$$\hat{f}_a(\mathbf{G}^{(\mathcal{D})}, \mathbf{W}, \mathbf{H}) = \left[\prod_{u=1}^{U}\prod_{d=1}^{D}\mathcal{N}(w_{ud}|\hat{m}_{u,d}^{a,w}, \hat{v}_{u,d}^{a,w})\right]\left[\prod_{d=1}^{D}\prod_{i=1}^{P}\mathcal{N}(h_{d,i}|\hat{m}_{d,i}^{a,h}, \hat{v}_{d,i}^{a,h})\right]$$
$$\left[\prod_{u=1}^{N}\prod_{j=1}^{M_u}\mathcal{N}(g_{u,z_{u,j}}|\hat{m}_{u,j}^{a,g}, \hat{v}_{u,j}^{a,g})\right]\hat{s}_a, \tag{13}$$

where $a = 1, \ldots, 4$ and $\hat{m}_{u,d}^{a,w}$, $\hat{v}_{u,d}^{a,w}$, $\hat{m}_{d,i}^{a,h}$, $\hat{v}_{d,i}^{a,h}$, $\hat{m}_{u,j}^{a,g}$, $\hat{v}_{u,j}^{a,g}$ and $\hat{s}_a$ are free parameters. Note that $\mathcal{Q}$ is the normalized product of $\hat{f}_1, \ldots, \hat{f}_4$. The first step of EP is to initialize $\hat{f}_1, \ldots, \hat{f}_4$ and $\mathcal{Q}$ to be uniform. After that, EP iteratively refines of $\hat{f}_1, \ldots, \hat{f}_4$ by minimizing the Kullback-Leibler (KL) divergence between the product of $\mathcal{Q}^{\backslash a}$ and $f_a$ and the product of $\mathcal{Q}^{\backslash a}$ and $\hat{f}_a$, where $\mathcal{Q}^{\backslash a}$ is the ratio between $\mathcal{Q}$ and $\hat{f}_a$. However, this does not perform well for refining $\hat{f}_2$; details on this problem can be found in Section 4 of the supplementary material and in [19]. For this factor we follow a VB approach. Instead of minimizing $\mathrm{KL}(\mathcal{Q}^{\backslash 2}f_2\|\mathcal{Q}^{\backslash 2}\hat{f}_2)$ with respect to the parameters of $\hat{f}_2$, we refine this approximate factor so that the reversed version of the KL divergence is minimized,

that is, we minimize $\text{KL}(\mathcal{Q}^{\backslash 2}\hat{f}_2 \| \mathcal{Q}^{\backslash 2}f_2)$. EP iteratively refines all the approximate factors until convergence. This method also approximates the predictive distribution (7). For this, we replace the exact posterior in (7) with $\mathcal{Q}$. Finally, EP can also approximate the normalization constant in (6) (the model evidence) as the integral of the product of all the approximate factors $\hat{f}_1, \ldots, \hat{f}_4$.

## 5.1 A sparse approximation to speed up computation

The cost of GPs is cubic in the number of function evaluations. In our case, refining $\hat{f}_3$ has cost $\mathcal{O}(DU^3)$, where $U$ is the number of users, and $D$ the number of shared latent functions. The cost of refining $\hat{f}_4$ is $\mathcal{O}(DP^3)$, where $P$ is the number of observed item pairs. These costs can be reduced by approximating $\mathbf{K}_{\text{users}}$ and $\mathbf{K}_{\text{items}}$ in (4) and (5). We use the FITC approximation [18]. Under this approximation, an $n \times n$ covariance matrix $\mathbf{K}$ generated by the evaluation of a covariance function at $n$ locations is approximated by $\mathbf{K}' = \mathbf{Q} + \text{diag}(\mathbf{K} - \mathbf{Q})$, where $\mathbf{Q} = \mathbf{K}_{nn_0}\mathbf{K}_{n_0 n_0}^{-1}\mathbf{K}_{nn_0}^{\text{T}}$, $\mathbf{K}_{n_0 n_0}$ is the $n_0 \times n_0$ matrix generated by the evaluation of the covariance function at all possible combinations of only $n_0 < n$ locations or pseudo-inputs and $\mathbf{K}_{nn_0}$ is the $n \times n_0$ matrix with the covariances between all possible combinations of original locations and pseudo-inputs. These approximations allow us to refine $\hat{f}_3$ and $\hat{f}_4$ in $\mathcal{O}(DU_0^2 U)$ and $\mathcal{O}(DP_0^2 P)$ operations, where $U_0$ and $P_0$ are the number of pseudo-inputs for the users and for the item pairs, respectively. A detailed description of the EP updates based on the FITC approximation is given in Section 4.4 of the supplementary material.

# 6 Experiments and Discussion

The performance of our collaborative preference model with the BALD active learning strategy is evaluated in a series of experiments with simulated and real-world data. The analyzed datasets include a) synthetic data generated from the probabilistic model assumed by the proposed multi-user method (Synthetic), b) a collection of user preferences on different movies (MovieLens), c) the number of votes obtained by different political parties in the 2010 UK general election (Election), d) preferences of users about different types of sushi (Sushi), and finally, e) information regarding the concentration of heavy metals in the Swiss Jura region (Jura). Section 6 in the supplementary material contains a detailed description of these datasets.

## 6.1 Comparison with other multi-user methods

**Alternative models.** Two versions of the proposed collaborative preference (CP) model are used. The first version (CPU) takes into account the available user features, as described in Section 3. The second version (CP) ignores these features by replacing $\mathbf{K}_{\text{users}}$ in (4) with the identity matrix. The first multi-user method we compare to is the approach of Birlitiu et al. (BI) [1]. This method does not use user features, and captures similarities between users with a hierarchical GP model. In particular, a common GP prior is assumed for the preference function of each user; using this prior the model learns the full GP posterior for each user. The second multi-user method is the technique of Bonilla et al. (BO) [2]. In this model there exists one high-dimensional function which depends on both the features of the two items to be compared and on the features of the user who makes the comparison. Relationships between users' behaviors are captured only via the user features. We implement BO and BI using the preference kernel and EP for approximate inference[1]. The computational costs of BO and BI are rather high; BO has cubic complexity in the *total* number of observations i.e. $\mathcal{O}((\sum_{u=1}^{U} M_u)^3)$, our model (CPU) has a significantly lower cost of $\mathcal{O}(D(U^3 + P^3))$ (before further speed-up from FITC). BI does not include user features, but learns $U$ GPs, so has complexity $\mathcal{O}(UP^3)$; the equivalent version of our model (CP) has cost $\mathcal{O}(NP + DP^3)$, which is lower because $D << U$. More details about BI and BO are given in sections 7 and 8 of the supplementary material. Finally, we consider a single user approach (SU) which fits a different GP classifier independently to the data of each user.

Table 1: Average test error with 100 users.

| Dataset | CPU | CP | BI | BO | SU |
|---|---|---|---|---|---|
| Synthetic | 0.162 | 0.180 | 0.175 | **0.157** | 0.226 |
| Sushi | 0.171 | 0.163 | **0.160** | 0.266 | 0.187 |
| MovieLens | 0.182 | **0.166** | 0.168 | 0.302 | 0.217 |
| Election | 0.199 | 0.123 | **0.077** | 0.401 | 0.300 |
| Jura | 0.159 | **0.153** | **0.153** | 0.254 | 0.181 |

Table 2: Training times (s) with 100 users.

| Dataset | CPU | CP | BI | BO | SU |
|---|---|---|---|---|---|
| Synthetic | 7.793 | 9.498 | 22.524 | 311.574 | 0.927 |
| Sushi | 5.694 | 4.307 | 20.028 | 215.136 | 0.817 |
| MovieLens | 5.313 | 4.013 | 19.366 | 69.048 | 0.604 |
| Election | 13.134 | 12.408 | 20.880 | 120.011 | 0.888 |
| Jura | 3.762 | 2.404 | 15.234 | 88.502 | 0.628 |

Table 3: Test error for each method and active learning strategy with at most 1000 users.

| Dataset | CPU-B | CPU-E | CPU-R | CP-B | CP-E | CP-R | SU-B | SU-E | SU-R |
|---|---|---|---|---|---|---|---|---|---|
| Synthetic | **0.135** | **0.135** | 0.139 | **0.153** | 0.160 | 0.173 | **0.249** | 0.259 | 0.268 |
| Sushi | **0.148** | 0.153 | 0.178 | **0.144** | 0.151 | 0.176 | **0.179** | 0.197 | 0.212 |
| MovieLens | **0.170** | 0.176 | 0.199 | **0.163** | 0.170 | 0.195 | **0.225** | 0.235 | 0.248 |
| Election | 0.202 | **0.158** | 0.224 | **0.097** | **0.093** | 0.151 | **0.332** | 0.346 | 0.338 |
| Jura | **0.143** | **0.141** | 0.168 | **0.138** | **0.138** | 0.169 | 0.176 | **0.166** | 0.197 |

**Experimental procedure.** Due to the high computational cost of BI and BO, to compare to these methods we must subsample the datasets, keeping only 100 users. The available data were split randomly into training and test sets of item pairs, where the training sets contain 20 pairs per user in Sushi, MovieLens and Election, 15 pairs in Jura and 30 in Synthetic. This was repeated 25 times to obtain statistically meaningful results. In CPU and CP, we selected the number of latent functions $D$ to be 20 (see Table 6.1). In general, the proposed models, CPU and CP, are robust to over-fitting and over-estimation of $D$ does not harm predictive performance. Note that the Synthetic dataset is generated using $D = 5$ and CPU and CP still obtain very good results using $D = 20$. This automatic pruning of unnecessary degrees of freedom seems to be common in methods based on variational Bayes [11]. We selected the kernel lengthscales to be equal to the median distance between feature vectors. This leads to good empirical performance for most methods. An exception is BO, where the kernel hyperparameters are tuned to some held-out data using automatic relevance determination. In our model, we can also estimate the kernel lengthscales by maximizing the EP approximation of the model evidence, as illustrated in Section 9 of the supplementary material. This alternative approach can be used when it is necessary to fine tune the lengthscale parameters to the data. In CPU we use $U_0 = 25$ pseudo inputs for approximating $\mathbf{K}_{\text{users}}$. These pseudo inputs are selected randomly from the set of available data points. Similarly, in CP and CPU, we use $P_0 = 25$ pseudo inputs for approximating $\mathbf{K}_{\text{items}}$, except in the Jura and Election datasets (which contain fewer items) where we use $P_0 = 15$. The results obtained are not sensitive to the number of pseudo inputs used, as long as the number is not excessively low.

**Results.** Average test errors are shown in Table 1. Those highlighted in bold are statistically different to those not highlighted (calculated using a paired $t$ test). Overall, CP and CPU outperform SU and BO, and breaks even with BI; the final result is notable as BI learns the full mean and covariance structure across all users, ours uses only a few latent dimensions, which provides the key to scaling to many more users. CP outperforms CPU in all cases except in the Synthetic dataset. In the real-world datasets, users with similar features do not seem to have similar preferences and so correlating behavior of users with similar features is detrimental. In this case, the unsupervised learning of similarities in user preferences is more useful for prediction than the user features. This also explains the poor overall results obtained by BO. Finally, running times in seconds are presented in Table 2. The entries for BO do not include the time spent by this method to tune the kernel hyper-parameters. CP and CPU are faster than BO and BI. The FITC approximation imposes a large multiplicative constant in the cost of CP and CPU so for larger datasets the gains are much larger.

## 6.2 Active learning on large datasets

Here we evaluate the performance of BALD, in particular, we compare CPU, CP, and SU using BALD (-B), Maximum Entropy Sampling (-E) and random sampling (-R). We now use all the available users from each dataset, with a maximum of 1000 users. For each user the available preference data are split randomly into training, pool and test sets with 5, 35 and 5 data points respectively in

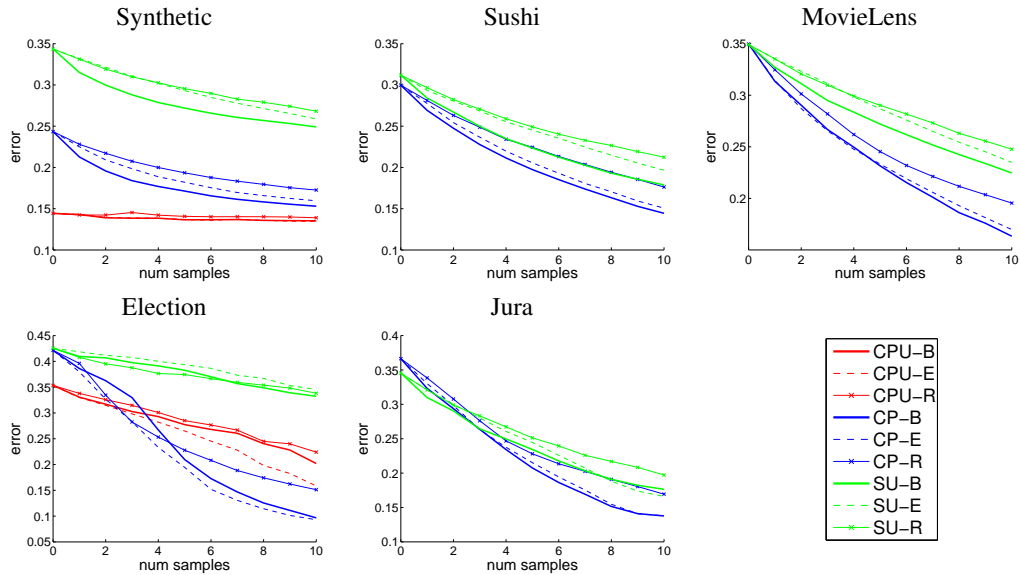

Figure 2: Average test error for CPU, CP and SU, using the strategies BALD (-B), entropy (-E) and random (-R) for active learning. For clarity, the curves for CPU are included only in the Synthetic and Election datasets. The complete plots can be found in Section 10 of the supplementary material.

Synthetic, Sushi and MovieLens, 3, 22 and 3 data points in Election and 3, 15 and 3 data points in Jura. Each method is fitted using the training sets and its performance is then evaluated on the corresponding test sets. After this, the most informative data point is identified in each of the pool sets. These data points are moved into the corresponding training sets and the process repeats until 10 of these active additions to the training sets have been completed. The entire process, including the dataset splitting is repeated 25 times. Figure 2 shows the learning curve for each method. For clarity, the curve for CPU is included only for the Synthetic and Election datasets; in the other datasets CPU is marginally outperformed by CP (see supplementary material, Section 10). Average errors after 10 queries from the pool set of each user are summarized in Table 3. For each model (CPU, CP and SU), the results of the best active learning strategy are highlighted in bold. The results of the best model/active learning strategy combination are underlined. Highlighted results are statistically significant with respect to non-highlighted results according to a paired $t$ test. BALD always outperforms random sampling and usually outperforms or obtains equivalent performance to MES. In particular, BALD significantly outperforms MES in 9 cases, while MES is better than BALD in only 2 cases.

# 7   Conclusions

We have proposed a multi-user model that combines collaborative filtering methods with GP binary preference modeling. We have shown that the task of learning user preferences can be recast as a particular case of binary classification with GPs when a covariance function called the preference kernel is used. We have also presented BALD, a novel active learning strategy for binary classification models with GPs. The proposed multi-user model with BALD performs favorably on simulated and real-world data against single-user methods and existing approaches for multi-user preference learning, whilst having significantly lower computational times than competing multi-user methods.

**Acknowledgements**

NH is a recipient of the Google Europe Fellowship in Statistical Machine Learning, and this research is supported in part by this Google Fellowship. JMH is supported by Infosys Labs, Infosys Limited.

## Footnotes

[1] Although this is not the same as the original implementations (sampling-based for BI, Laplace approximation for BO), the preference kernel and EP are likely to augment the performance of these algorithms, and provides the fairest comparison of the underlying models.

# References

[1] A. Birlutiu, P. Groot, and T. Heskes. Multi-task preference learning with an application to hearing aid personalization. *Neurocomputing*, 73(79):1177 – 1185, 2010.

[2] Edwin V. Bonilla, Shengbo Guo, and Scott Sanner. Gaussian process preference elicitation. In *Advances in Neural Information Processing Systems 23*, pages 262–270, 2010.

[3] E. Brochu, N. de Freitas, and A. Ghosh. Active preference learning with discrete choice data. *Advances in Neural Information Processing Systems 20*, 20:409–416, 2007.

[4] W. Chu and Z. Ghahramani. Preference learning with Gaussian processes. In *Proceedings of the 22nd international conference on Machine learning*, pages 137–144, 2005.

[5] M. De Gemmis, L. Iaquinta, P. Lops, C. Musto, F. Narducci, and G. Semeraro. Preference learning in recommender systems. In *ECML/PKDD-09 Workshop on Preference Learning*, 2009.

[6] J. Fürnkranz and E. Hüllermeier. *Preference learning*. Springer-Verlag New York Inc, 2010.

[7] Z. Ghahramani and M. J. Beal. *Advanced Mean Field Method—Theory and Practice*, chapter Graphical models and variational methods, pages 161–177. 2001.

[8] B. Krishnapuram, D. Williams, Y. Xue, A. Hartemink, L. Carin, and M. Figueiredo. On semi-supervised classification. In *Advances in neural information processing systems 17*, pages 721–728, 2004.

[9] N.D. Lawrence, M. Seeger, and R. Herbrich. Fast sparse gaussian process methods: The informative vector machine. *Advances in Neural Information Processing Systems 15*, 15:609–616, 2002.

[10] D.V. Lindley. On a measure of the information provided by an experiment. *The Annals of Mathematical Statistics*, 27(4):986–1005, 1956.

[11] D. J. C. MacKay. Local minima, symmetry-breaking, and model pruning in variational free energy minimization. Available at http://www.inference.phy.cam.ac.uk/mackay/minima.pdf, 2001.

[12] D.J.C. MacKay. Information-based objective functions for active data selection. *Neural computation*, 4(4):590–604, 1992.

[13] T. Minka and J. Lafferty. Expectation-propagation for the generative aspect model. In *Proceedings of the Eighteenth conference on Uncertainty in artificial intelligence*, pages 352–359, 2002.

[14] Tom Minka. *A family of algorithms for approximate Bayesian inference*. PhD thesis, MIT, 2001.

[15] Hannes Nickisch and Carl Edward Rasmussen. Approximations for binary Gaussian process classification. *The Journal of Machine Learning Research*, 9:2035–2078, 2008.

[16] T. Raiko, A. Ilin, and K. Juha. Principal component analysis for large scale problems with lots of missing values. In Joost Kok, Jacek Koronacki, Raomon Mantaras, Stan Matwin, Dunja Mladenic, and Andrzej Skowron, editors, *Machine Learning: ECML 2007*, volume 4701 of *Lecture Notes in Computer Science*, pages 691–698. Springer Berlin / Heidelberg, 2007.

[17] P. Sebastiani and H.P. Wynn. Maximum entropy sampling and optimal Bayesian experimental design. *Journal of the Royal Statistical Society. Series B (Statistical Methodology)*, 62(1):145–157, 2000.

[18] E. Snelson and Z. Ghahramani. Sparse gaussian processes using pseudo-inputs. In *Advances in Neural Information Processing Systems 18*, 2005.

[19] D. H. Stern, R. Herbrich, and T. Graepel. Matchbox: large scale online bayesian recommendations. In *Proceedings of the 18th international conference on World wide web*, pages 111–120, 2009.

